# Back Propagation is Sensitive to Initial Conditions

**John F. Kolen**     **Jordan B. Pollack**
Laboratory for Artificial Intelligence Research
The Ohio State University
Columbus, OH 43210, USA
kolen-j@cis.ohio-state.edu
pollack@cis.ohio-state.edu

## Abstract

This paper explores the effect of initial weight selection on feed-forward networks learning simple functions with the back-propagation technique. We first demonstrate, through the use of Monte Carlo techniques, that the magnitude of the initial condition vector (in weight space) is a very significant parameter in convergence time variability. In order to further understand this result, additional deterministic experiments were performed. The results of these experiments demonstrate the extreme sensitivity of back propagation to initial weight configuration.

## 1 INTRODUCTION

Back Propagation (Rumelhart *et al.*, 1986) is the network training method of choice for many neural network projects, and for good reason. Like other weak methods, it is simple to implement, faster than many other "general" approaches, well-tested by the field, and easy to mold (with domain knowledge encoded in the learning environment) into very specific and efficient algorithms.

Rumelhart *et al.* made a confident statement: for many tasks, "the network rarely gets stuck in poor local minima that are significantly worse than the global minima."(p. 536) According to them, initial weights of exactly 0 cannot be used, since symmetries in the environment are not sufficient to break symmetries in initial weights. Since their paper was published, the convention in the field has been to choose initial weights with a uniform distribution between plus and minus $\rho$, usually set to 0.5 or less.

The convergence claim was based solely upon their empirical experience with the back propagation technique. Since then, Minsky & Papert (1988) have argued that there exists no proof of convergence for the technique, and several researchers (*e.g.* Judd 1988) have found that the convergence time must be related to the difficulty of the problem, otherwise an unsolved computer science question ($P \overset{?}{=} NP$) would finally be answered. We do not wish to make claims about convergence of the technique in the limit (with vanishing step-

size), or the relationship between task and performance, but wish to talk about a pervasive behavior of the technique which has gone unnoticed for several years: the sensitivity of back propagation to initial conditions.

## 2     THE MONTE-CARLO EXPERIMENT

Initially, we performed empirical studies to determine the effect of learning rate, momentum rate, and the range of initial weights on t-convergence (Kolen and Goel, to appear). We use the term *t-convergence* to refer to whether or not a network, starting at a precise initial configuration, could learn to separate the input patterns according to a boolean function (correct outputs above or below 0.5) within $t$ epochs. The experiment consisted of training a 2-2-1 network on exclusive-or while varying three independent variables in 114 combinations: learning rate, $\eta$, equal to 1.0 or 2.0; momentum rate, $\alpha$, equal to 0.0, 0.5, or 0.9; and initial weight range, $\rho$, equal to 0.1 to 0.9 in 0.1 increments, and 1.0 to 10.0 in 1.0 increments. Each combination of parameters was used to initialize and train a number of networks.[1] Figure 1 plots the percentage of t-convergent (where $t \cong 50,000$ epochs of 4 presentations) initial conditions for the 2-2-1 network trained on the exclusive-or problem. From the figure we thus conclude the choice of $\rho \le 0.5$ is more than a convenient symmetry-breaking default, but is quite necessary to obtain low levels of nonconvergent behavior.

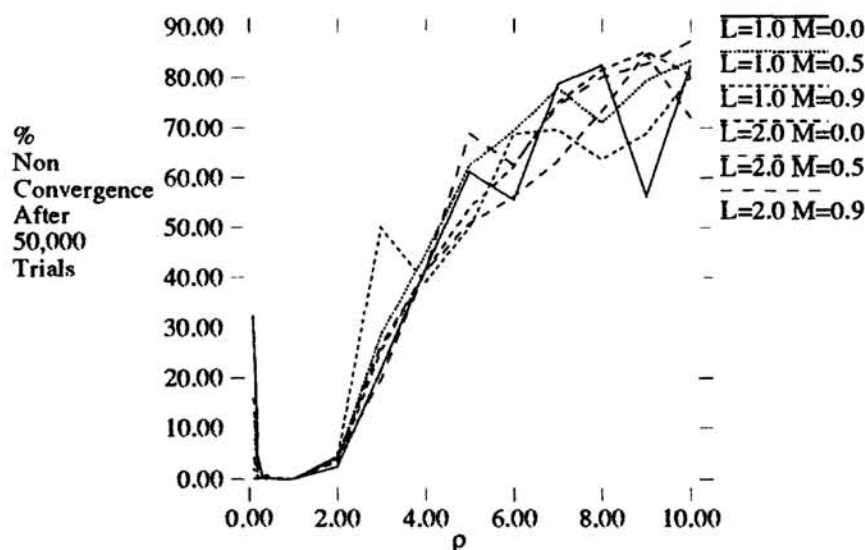

Figure 1: Percentage T-Convergence vs. Initial Weight Range

## 3     SCENES FROM EXCLUSIVE-OR

Why do networks exhibit the behavior illustrated in Figure 1? While some might argue that very high initial weights (i.e. $\rho > 10.0$) lead to very long convergence times since the derivative of the semi-linear sigmoid function is effectively zero for large weights, this

---

1. Numbers ranged from 8 to 8355, depending on availability of computational resources. Those data points calculated with small samples were usually surrounded by data points with larger samples.

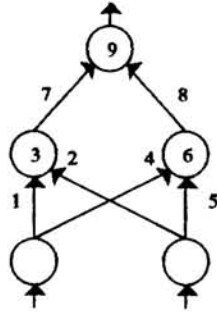

Figure 2:
(Schematic Network)

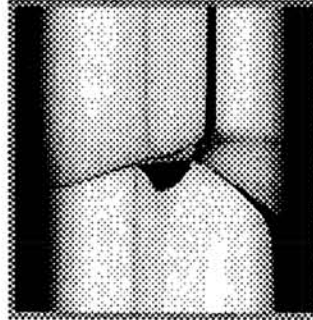

Figure 3:
(-5-3+3+6Y-1-6+7X)
η=3.25 α=0.40

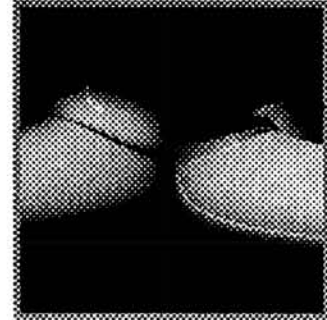

Figure 4:
(+4-7+6+0-3Y+1X+1)
η=2.75 α=0.00

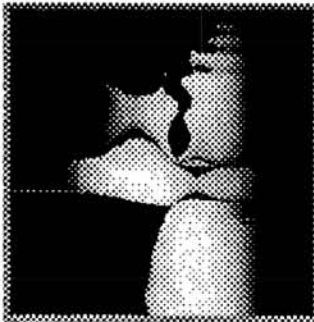

Figure 5:
(-5+5+1-6+3XY+8+3)
η=2.75 α=0.80

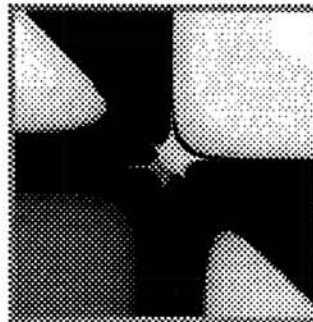

Figure 6:
(YX-3+6+8+3+1+7-3)
η=3.25 α=0.00

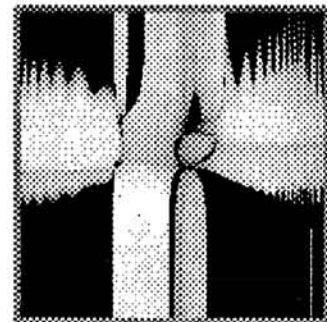

Figure 7:
(Y+3-9-2+6+7-3X+7)
η=3.25 α=0.60

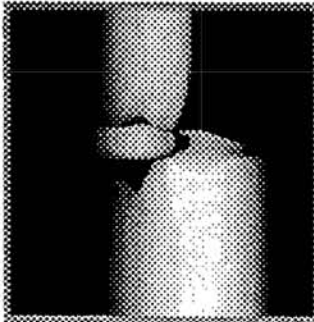

Figure 8:
(-6-4XY-6-6+9-4-9)
η=3.00 α=0.50

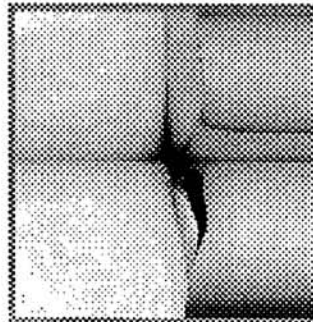

Figure 9:
(-2+1+9-1X-3+8Y-4)
η=2.75 α=0.20

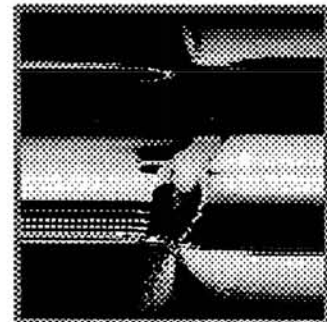

Figure 10:
(+1+8-3-6X-1+1+8Y)
η=3.50 α=0.90

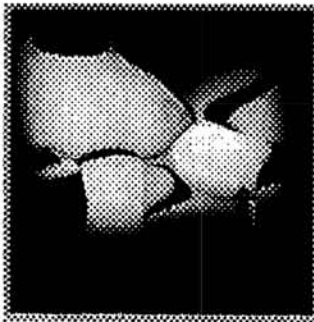

Figure 11:
(+7+4-9-9-5Y-3+9X)
η=3.00 α=0.70

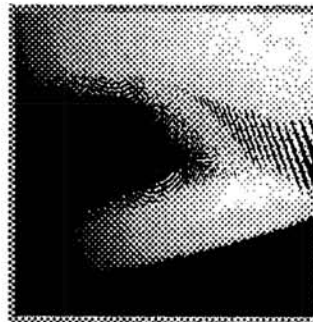

Figure 12:
(-9.0,-1.8)
step 0.018

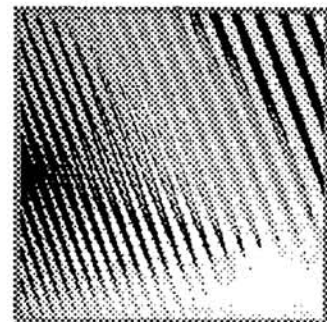

Figure 13:
(-6.966,-0.500)
step 0.004

does not explain the fact that when $\rho$ is between 2.0 and 4.0, the non-t-convergence rate varies from 5 to 50 percent.

Thus, we decided to utilize a more deterministic approach for eliciting the structure of initial conditions giving rise to t-convergence. Unfortunately, most networks have many weights, and thus many dimensions in initial-condition space. We can, however, examine 2-dimensional slices through the space in great detail. A slice is specified by an origin and two orthogonal directions (the X and Y axes). In the figures below, we vary the initial weights regularly throughout the plane formed by the axes (with the origin in the lower left-hand corner) and collect the results of running back-propagation to a particular time limit for each initial condition. The map is displayed with grey-level linearly related to time of convergence: black meaning not t-convergent and white representing the fastest convergence time in the picture. Figure 2 is a schematic representation of the networks used in this and the following experiment. The numbers on the links and in the nodes will be used for identification purposes. Figures 3 through 11 show several interesting "slices" of the the initial condition space for 2-2-1 networks trained on **exclusive-or**. Each slice is compactly identified by its 9-dimensional weight vector and associated learning/ momentum rates. For instance, the vector (-3+2+7-4X+5-2-6Y) describes a network with an initial weight of -0.3 between the left hidden unit and the left input unit. Likewise, "+5" in the sixth position represents an initial bias of 0.5 to the right hidden unit. The letters "X" and "Y" indicate that the corresponding weight is varied along the X- or Y- axis from -10.0 to +10.0 in steps of 0.1. All the figures in this paper contain the results of 40,000 runs of back-propagation (*i.e.* 200 pixels by 200 pixels) for up to 200 epochs (where an epoch consists of 4 training examples).

Figures 12 and 13 present a closer look at the sensitivity of back-propagation to initial conditions. These figures zoom into a complex region of Figure 11; the captions list the location of the origin and step size used to generate each picture.

Sensitivity behavior can also be demonstrated with even simpler functions. Take the case of a 2-2-1 network learning the **or** function. Figure 14 shows the effect of learning "or" on networks (+5+5-1X+5-1Y+3-1) and varying weights 4 (X-axis) and 7 (Y-axis) from -20.0 to 20.0 in steps of 0.2. Figure 15 shows the same region, except that it partitions the display according to equivalent solution networks after t-convergence (200 epoch limit), rather than the time to convergence. Two networks are considered equivalent[2] if their weights have the same sign. Since there are 9 weights, there are 512 ($2 sup 9$) possible network equivalence classes. Figures 16 through 25 show successive zooms into the central swirl identified by the XY coordinate of the lower-left corner and pixel step size. After 200 iterations, the resulting networks could be partitioned into 37 (both convergent and nonconvergent) classes. Obviously, the smooth behavior of the t-convergence plots can be deceiving, since two initial conditions, arbitrarily alike, can obtain quite different final network configuration.

Note the triangles appearing in Figures 19, 21, 23 and the mosaic in Figure 25 corresponding to the area which did not converge in 200 iterations in Figure 24. The triangular boundaries are similar to fractal structures generated under iterated function systems (Barnsley 1988): in this case, the iterated function is the back propagation

---

2. For rendering purposes only. It is extremely difficult to know precisely the equivalence classes of solutions, so we approximated.

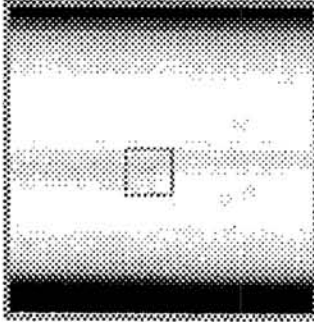

Figure 14 :
(-20.00000, -20.00000)
Step 0.200000

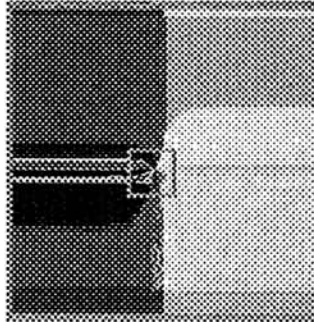

Figure 15 :
Solution Networks

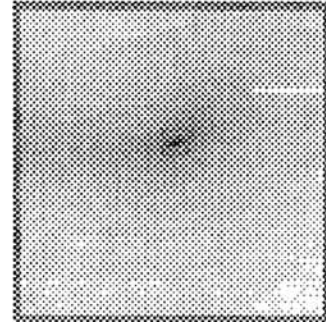

Figure 16 :
(-4.500000, -4.500000)
Step 0.030000

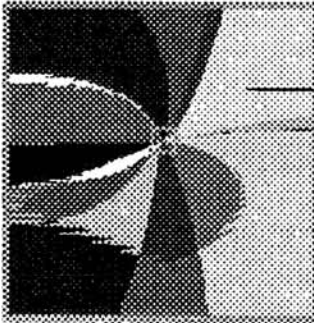

Figure 17 :
Solution Networks

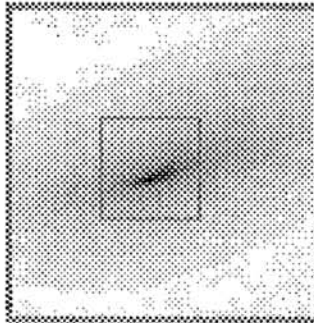

Figure 18 :
(-1.680000, -1.350000)
Step 0.002400

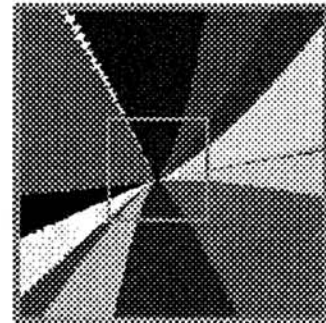

Figure 19 :
Solution Networks

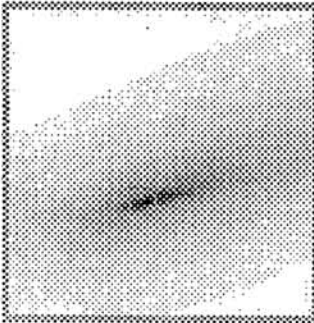

Figure 20 :
(-1.536000, -1.197000)
Step 0.000780

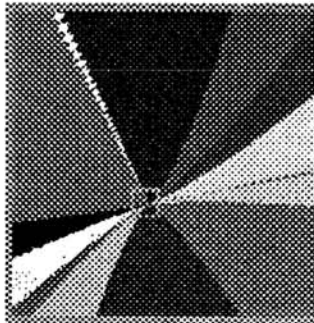

Figure 21 :
Solution Networks

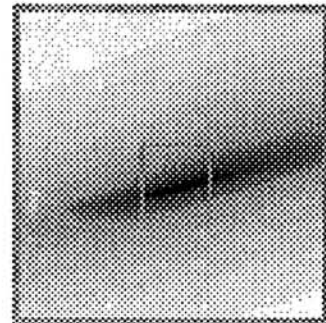

Figure 22 :
(-1.472820, -1.145520)
Step 0.000070

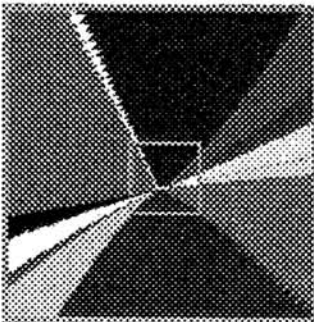

Figure 23 :
Solution Networks

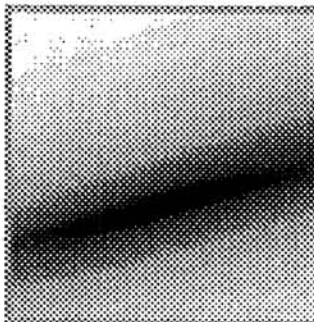

Figure 24 :
(-1.467150, -1.140760)
Step 0.000016

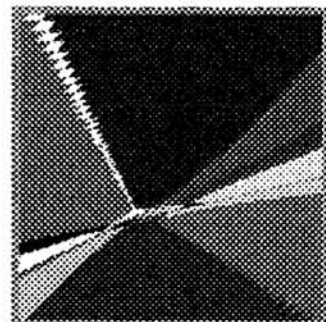

Figure 25 :
Solution Networks

|            | Figure 26     | Figure 28     | Figure 27 Figure 29 Figure 30 |
|------------|---------------|---------------|-------------------------------|
| Weight 1   | -0.34959000   | -0.34959000   | -0.34959000                   |
| Weight 2   | 0.00560000    | 0.00560000    | 0.00560000                    |
| Weight 3   | -0.26338813   | 0.39881098    | 0.65060705                    |
| Weight 4   | 0.75501968    | -0.16718577   | 0.75501968                    |
| Weight 5   | 0.47040862    | -0.28598450   | 0.91281711                    |
| Weight 6   | -0.18438011   | -0.18438011   | -0.19279729                   |
| Weight 7   | 0.46700363    | -0.06778983   | 0.56181073                    |
| Weight 8   | -0.48619500   | 0.66061292    | 0.20220653                    |
| Weight 9   | 0.62821201    | -0.39539510   | 0.11201949                    |
| Weight 10  | -0.90039973   | 0.55021922    | 0.67401200                    |
| Weight 11  | 0.48940201    | 0.35141364    | -0.54978875                   |
| Weight 12  | -0.70239312   | -0.17438740   | -0.69839197                   |
| Weight 13  | -0.95838741   | -0.07619988   | -0.19659844                   |
| Weight 14  | 0.46940394    | 0.88460041    | 0.89221204                    |
| Weight 15  | -0.73719884   | 0.67141031    | -0.56879740                   |
| Weight 16  | 0.96140103    | -0.10578894   | 0.20201484                    |

Table 1: Network Weights for Figures 26 through 30

learning method. We propose that these fractal-like boundaries arise in back-propagation due to the existence of multiple solutions (attractors), the non-zero learning parameters, and the non-linear deterministic nature of the gradient descent approach. When more than one hidden unit is utilized, or when an environment has internal symmetry or is very underconstrained, then there will be multiple attractors corresponding to the large number of hidden-unit permutations which form equivalence classes of functionality. As the number of solutions available to the gradient descent method increases, the more complicated the non-local interactions between them. This explains the puzzling result that several researchers have noted, that as more hidden units are added, instead of speeding up, back-propagation slows down (*e.g.* Lippman and Gold, 1987). Rather than a hill-climbing metaphor with local peaks to get stuck on, we should instead think of a many-body metaphor: The existence of many bodies does not imply that a particle will take a simple path to land on one. From this view, we see that Rumelhart *et al.*'s claim of back-propagation usually converging is due to a very tight focus inside the "eye of the storm".

Could learning and momentum rates also be involved in the storm? Such a question prompted another study, this time focused on the interaction of learning and momentum rates. Rather than alter the initial weights of a set of networks, we varied the learning rate along the X axis and momentum rate along the Y axis. Figures 26, 27, and 28 were produced by training a 3-3-1 network on 3-bit parity until t-convergence (250 epoch limit). Table 1 lists the initial weights of the networks trained in Figures 26 through 31. Examination of the fuzzy area in Figure 26 shows how small changes in learning and/or momentum rate can drasticly affect t-convergence (Figures 30 and 31).

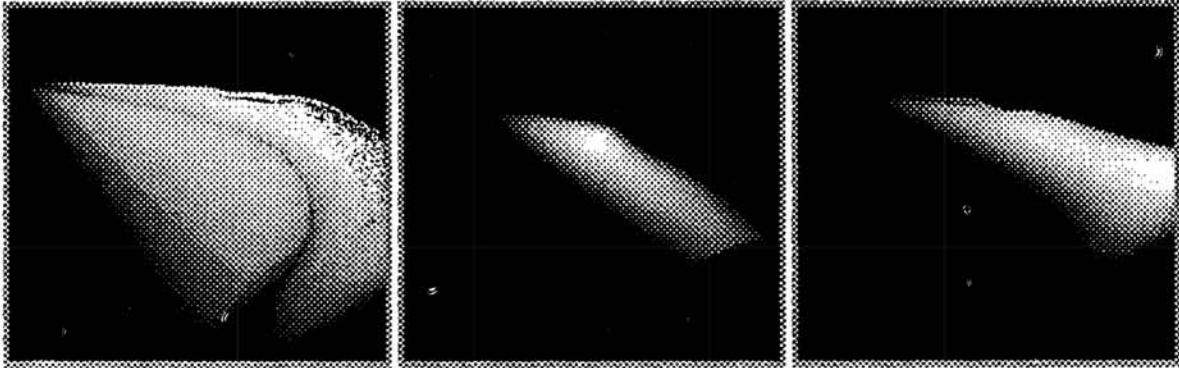

Figure 26:
η=(0.0,4.0) α=(0.0,1.25)

Figure 27:
η=(0.0,4.0) α=(0.0,1.25)

Figure 28:
η=(0.0,4.0) α=(0.0,1.25)

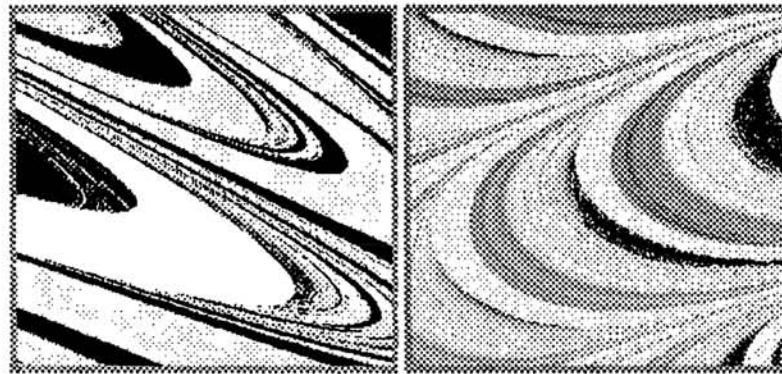

Figure 29:
η=(3.456,3.504)
α=(0.835,0.840)

Figure 30:
η=(3.84,3.936)
α=(0.59,0.62)

## 4   DISCUSSION

Chaotic behavior has been carefully circumvented by many neural network researchers (through the choice of symmetric weights by Hopfield (1982), for example), but has been reported in increasing frequency over the past few years (*e.g.* Kurten and Clark, 1986). Connectionists, who use neural models for cognitive modeling, disregard these reports of extreme non-linear behavior in spite of common knowledge that non-linearity is what enables network models to perform non-trivial computations in the first place, All work to date has noticed various forms of chaos in network dynamics, but not in learning dynamics. Even if back-propagation is shown to be non-chaotic in the limit, this still does not preclude the existance of fractal boundaries between attractor basins since other non-chaotic non-linear systems produce such boundaries (i.e. forced pendulums with two attractors (D'Humieres *et al.*, 1982))

What does this mean to the back-propagation community? From an engineering applications standpoint, where only the solution matters, nothing at all. When an optimal set of weights for a particular problem is discovered, it can be reproduced through digital means. From a scientific standpoint, however, this sensitivity to initial conditions demands that neural network *learning* results must be specially treated to guarantee replicability. When theoretical claims are made (from experience) regarding the power of an adaptive

network to model some phenomena, or when claims are made regarding the similarity between psychological data and network performance, **the initial conditions for the network need to be precisely specified or filed in a public scientific database.**

What about the future of back-propagation? We remain neutral on the issue of its ultimate convergence, but our result points to a few directions for improved methods. Since the slowdown occurs as a result of global influences of multiple solutions, an algorithm for first factoring the symmetry out of both network and training environment (*e.g.* domain knowledge) may be helpful. Furthermore, it may also turn out that search methods which harness "strange attractors" ergodically guaranteed to come arbitrarily close to somesubset of solutions might work better than methods based on strict gradient descent. Finally, we view this result as strong impetus to discover how to exploit the information-creative aspects of non-linear dynamical systems for future models of cognition (Pollack 1989).

## Acknowledgements

This work was supported by Office of Naval Research grant number N00014-89-J1200. Substantial free use of over 200 Sun workstations was generously provided by our department.

## References

M. Barnsley,*Fractals Everywhere*, Academic Press, San Diego, CA, (1988).

J. J. Hopfield, "Neural Networks and Physical Systems with Emergent Collective Computational Abilities", *Proceedings US National Academy of Science*, **79**:2554-2558, (1982).

D. D'Humieres, M. R. Beasley, B. A. Huberman, and A. Libchaber, "Chaotic States and Routes to Chaos in the Forced Pendulum", *Physical Review A*, **26**:3483-96, (1982).

S. Judd, "Learning in Networks is Hard", *Journal of Complexity*, **4**:177-192, (1988).

J. Kolen and A. Goel, "Learning in Parallel Distributed Processing Networks: Computational Complexity and Information Content", *IEEE Transactions on Systems, Man, and Cybernetics*, in press.

K. E. Kürten and J. W. Clark, "Chaos in Neural Networks", *Physics Letters*, **114A**, 413-418, (1986).

R. P. Lippman and B. Gold, "Neural Classifiers Useful for Speech Recognition", In *1st International Conference on Neural Networks* ,IEEE, IV:417-426, (1987).

M. L. Minsky and S. A. Papert, *Perceptrons*. MIT Press, (1988).

J. B. Pollack, "Implications of Recursive Auto Associative Memories", In *Advances in Neural Information Processing Systems*. (ed. D. Touretzky) pp 527-536, Morgan Kaufman, San Mateo, (1989) .

D. E. Rumelhart, G. E. Hinton, and R. J. Williams, "Learning Representation by Back-Propagating Errors", *Nature*, **323**:533-536, (1986).